# Using A Translation-Invariant Neural Network To Diagnose Heart Arrhythmia

**Susan Ciarrocca Lee**
The Johns Hopkins University
Applied Physics Laboratory
Laurel, Maryland 20707

## ABSTRACT

Distinctive electrocardiogram (ECG) patterns are created when the heart is beating normally and when a dangerous arrhythmia is present. Some devices which monitor the ECG and react to arrhythmias parameterize the ECG signal and make a diagnosis based on the parameters. The author discusses the use of a neural network to classify the ECG signals directly, without parameterization. The input to such a network must be translation-invariant, since the distinctive features of the ECG may appear anywhere in an arbritrarily-chosen ECG segment. The input must also be insensitive to the episode-to-episode and patient-to-patient variability in the rhythm pattern.

## 1 INTRODUCTION

Figure 1 shows internally-recorded transcardiac ECG signals for one patient. The top trace is an example of normal sinus rhythm (NSR). The others are examples of two arrhythmias: ventricular tachycardia (VT) and ventricular fibrillation (VF). Visually, the patterns are quite distinctive. Two problems make recognition of these patterns with a neural net interesting.

The first problem is illustrated in Figure 2. All traces in Figure 2 are one second samples of NSR, but the location of the QRS complex relative to the start of the sample is shifted. Ideally, one would like a neural network to recognize each of these presentations as NSR, without preprocessing the data to "center" it. The second problem can be discerned by examining the two VT traces in Figure 1. Although quite similar, the two patterns are not exactly the same. Substantial variation in signal shape and repetition rate for NSR and VT (VF is inherently random) can be expected, even among rhythms generated by a single patient. Patient-to-patient variations are even greater. The neural

network must ignore variations within rhythm types, while retaining the distinctions between rhythms. This paper discusses a simple transformation of the ECG time series input which is both translation-invariant and fairly insensitive to rate and shape changes within rhythm types.

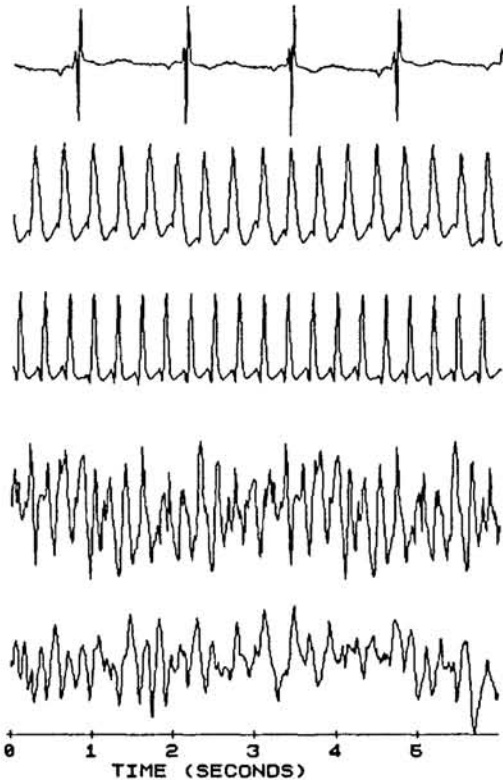

Figure 1: ECG Rhythm Examples

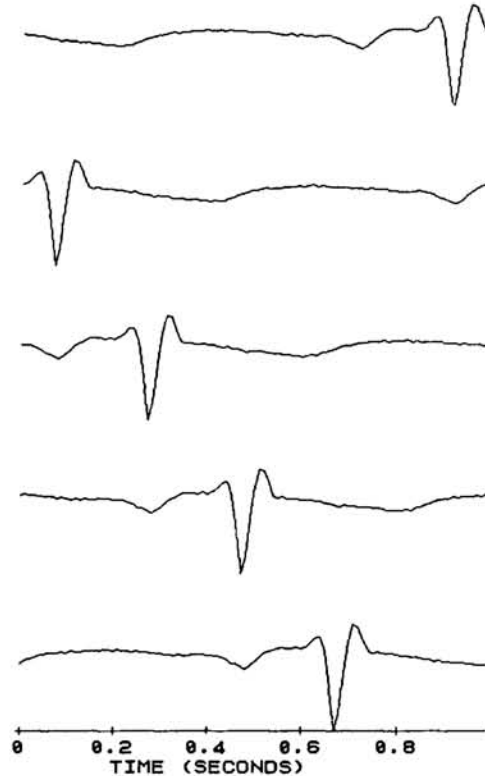

Figure 2: Five Examples of NSR

## 2  DISCUSSION

If test input to a first order neural network is rescaled, rotated, or translated with respect to the training data, it generally will not be recognized. A second or higher order network can be made invariant to these transformations by constraining the weights to meet certain requirements[Giles, 1988]. The input to the jth hidden unit in a second order network with N inputs is:

$$\sum_{i=1}^{N} w_{ij}x_i \; + \; \sum_{i=1}^{N-1} \sum_{k=1}^{N-i} w_{(i,i+k)j}x_i x_{i+k} \tag{1}$$

Translation invariance is introduced by constraining the weights on the first order inputs to be independent of input position, and the second order weights to depend only on the difference between indices (k), rather than on the index pairs (i,i+k)[Giles, 1988]. Rewriting equation (1) with these constraints gives:

$$w_j \sum_{i=1}^{N} x_i + \sum_{k=1}^{N-1} w_{kj} \sum_{i=1}^{N-k} x_i x_{i+k} \qquad (2)$$

This is equivalent to a first order neural network where the original inputs, $x_i$, have been replaced by new inputs, $y_i$, consisting of the following sums:

$$y_0 = \sum_{i=1}^{N} x_i \ , \qquad y_k = \sum_{i=1}^{N-k} x_i x_{i+k} \ , \ k=1,2,...,N-1 \qquad (3)$$

While a network with inputs in the form of equation (3) is translation invariant, it is quite sensitive to shape and rate variations in the ECG input data. For ECG recognition, a better function to compute is:

$$y_0 = \sum_{i=1}^{N} ABS(x_i) \ , \qquad y_k = \sum_{i=1}^{N-k} ABS(x_i - x_{i+k}) \ , \qquad k=1,2,...,N-1 \qquad (4)$$

Both equations (3) and (4) produce translation-invariant outputs, as long as the input time series contains a "shape" which occupies only part of the input window, for example, the single cycle of the sine function in Figure 3a. A periodic time series, like the sine wave in Figure 3b, will not produce a truly translation-invariant output. Fortunately, the translation sensitivity introduced by applying equations (3) or (4) to periodic time series is small for small k, and only becomes important when k becomes large. One can see this by considering the extreme case, when k=N-1, and the final "sum" in equation (4) becomes the absolute value of the difference between the first and the last point in the input time series; clearly, this value will vary as the sine wave in Figure 3b is moved through the input window. If the upper limit on the sum over k gets no larger than N/2,

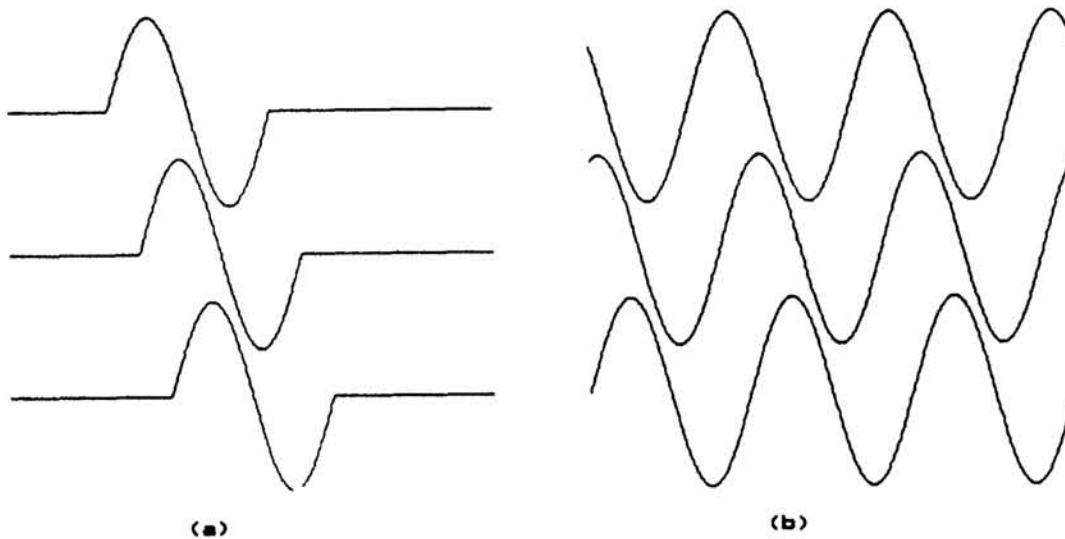

(a)                                    (b)

**Figure 3:** Examples of signals which will (a) and will not (b) have invariant transforms

equations (3) and (4) provide a neural network input which is nearly translation-invariant for realistic time series. Additionally, the output of equation (4) can be used to discriminate among NSR, VT, and VF, but is not unduly sensitive to variations within each rhythm type.

The ECG signals used in this experiment were drawn from a data set of internally recorded transcardiac ECG signals digitized at 100 Hz. The data set comprised 203 10-45 second segments obtained from 52 different patients. At least one segment of NSR and one segment of an arrhythmia was available for each patient. In addition, an "exercise" NSR at 150 BPM was artificially constructed by cutting baseline out of the natural resting NSR segment. Arrhythmia detection systems which parameterize the ECG can have difficulty distinguishing high rate NSR's from slow arrhythmias.

To obtain a training data set for the neural network, short pieces were extracted from the original rhythm segments. Since the rhythms are basically periodic, it was possible to chose the endpoints so that the short, extracted piece could be be repeated to produce a facsimile of the original signal. The upper trace in Figure 4 shows an original VT segment. The boxed area is the extracted piece. The lower trace shows the extracted piece chained end-to-end to construct a segment as long as the original. The segments

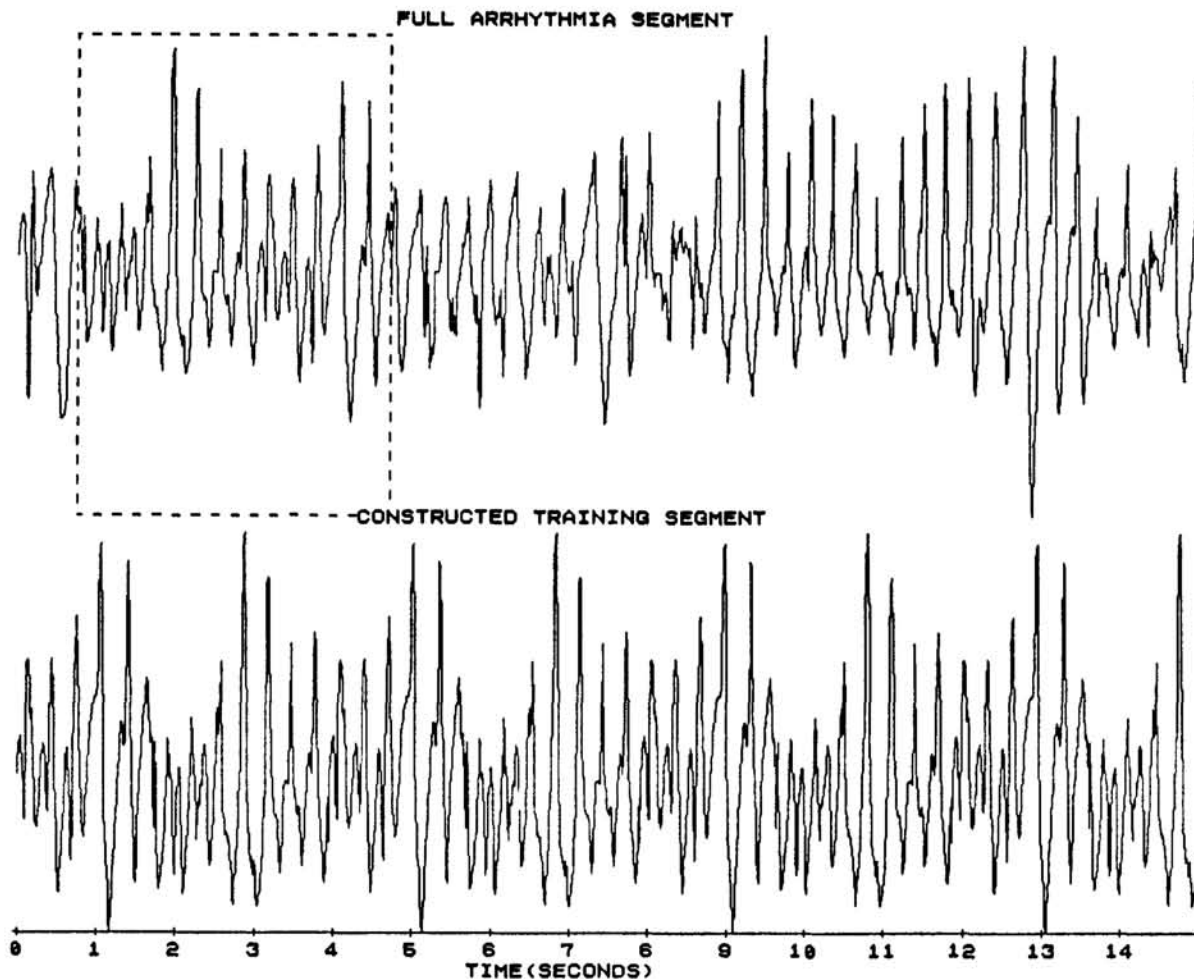

**Figure 4:** Original and Artificially-Constructed Training Segments

constructed from the short, extracted pieces were used as training input. Typically, the training data segment contained less than 25% of the original data.

The length of the input window was arbitrarily set at 1.35 seconds (135 points); by choosing this window, all NSR inputs were guaranteed to include at least one QRS complex. The upper limit on the sum over k in equation (4) was set to 50. The resulting 51 inputs were presented to a standard back propagation network with seven hidden units and four outputs. Although one output is sufficient to discriminate between NSR and an arrhythmia, the networks were trained to differentiate among two types of VT (generally distinguished by rate), and VF as well.

A separate training set was constructed and a separate network was trained for each patient. The weights thus derived for a given patient were then tested on that patient's original rhythm segments. To test the translation invariance of the network, every possible presentation of an input rhythm segment was tested. To do this, a sliding window of 135 points was moved through the input data stream one point (1/100th of a second) at a time. At each point, the output of equation (4) (appropriately normalized) was presented to the network, and the resulting diagnosis recorded.

## 3  RESULTS

A percentage of correct diagnoses was calculated for each segment of data. For a segment T seconds long, there are 100x(T-1.35) different presentations of the rhythm. Presentations which included countershock, burst pacing, gain changes on the recording equipment, post-shock rhythms, etc. were excluded, since the network had not been trained to recognize these phenomena. The percentage correct was then calculated for the remaining presentations as:

100x(Number of correct diagnoses)/(Number of presentations)

The percentage of correct diagnoses for each patient was calculated similarly, except that all segments for a particular patient were included in the count. Table 1 presents these results.

**Table  1:** Results

|                      | Patients | Segments |
|----------------------|----------|----------|
| 100% Correct         | 29       | 163      |
| 99%-90% Correct      | 19       | 23       |
| 90%-80% Correct      | 3        | 6        |
| 80%-70% Correct      | 0        | 4        |
| <70% Correct         | 0        | 1        |
| Could Not Be Trained | 1        | 6        |
| Total                | 52       | 203      |

The network could not be trained for one patient. This patient had two arrhythmia segments, one identified as VT and the other as VF. Visually, the two traces were extremely similiar; after twenty thousand iterations, the network could not distinguish them. The network could certainly have been trained to distinguish between NSR and those two rhythms, but this was not attempted.

The number of segments for which all possible presentations of the rhythm were diagnosed correctly clearly establishes the translation invariance of the input. The network was also quite successful in distinguishing among NSR and various arrhythmias. Unfortunately, for application in inplantable defibrillators or even critical care monitoring, the network must be more nearly perfect.

The errors the network made could be separated into two broad classes. First, short segments of very erratic arrhythmias were misdiagnosed as NSR. Figure 5 illustrates this type of error. The error occurs because NSR is mainly characterized by a lack of correlation. Typically, the misdiagnosed segment is quite short, 1 second or less. This type of error might be avoided by using longer (longer than 1.35 second) input windows which could bridge the erratic segments. Also, a more responsive automatic gain control on the signal might help, since the erratic segments generally had a smaller amplitude

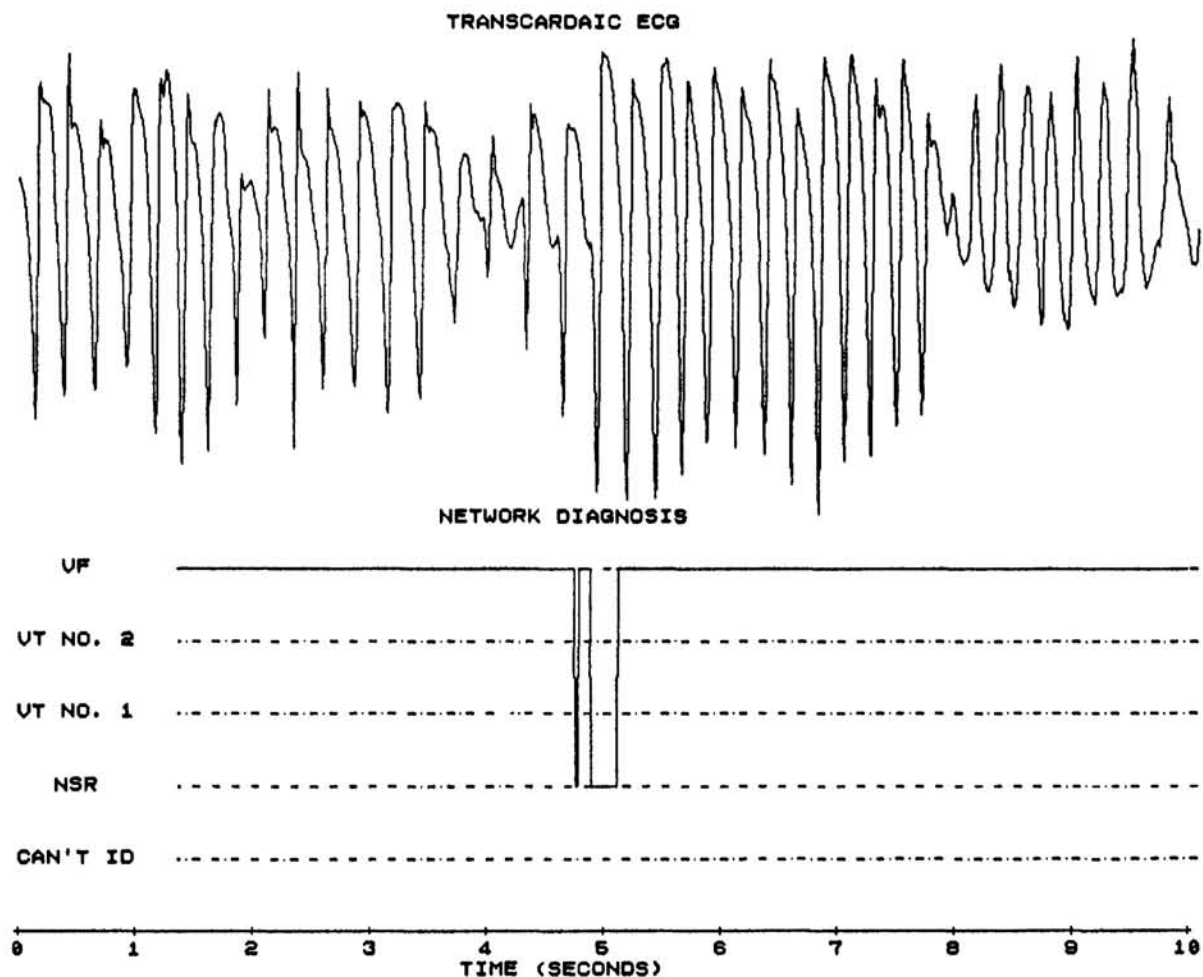

**Figure 5:** Ventricular Fibrillation Segment Misdiagnosed as NSR

than the surrounding segments. The network response to input windows containing large shifts in the amplitude of the input signal (for example, countershock and gain changes) was usually NSR.

The second class of errors occurred when the network misdiagnosed rhythms which were not included in the training set. For example, one patient had a few beats of a very slow VT in his NSR segment. This slow VT was not extracted for training. Only a fast (200 BPM) VT and VF were presented to this network as possible arrhythmias. Consequently, during testing, the network identified the slow VT as NSR. The network did identify some rhythms it was not trained on, but only if these rhythms did not vary too much from the training rhythms. Generally, the rate of the "unknown" rhythm had to be within 20 BPM of a training rhythm to be recognized. Morphology is also important, in that very regular rhythms, such as the top trace in Figure 6, and noisier rhythms, like the bottom trace, appear quite different to the network.

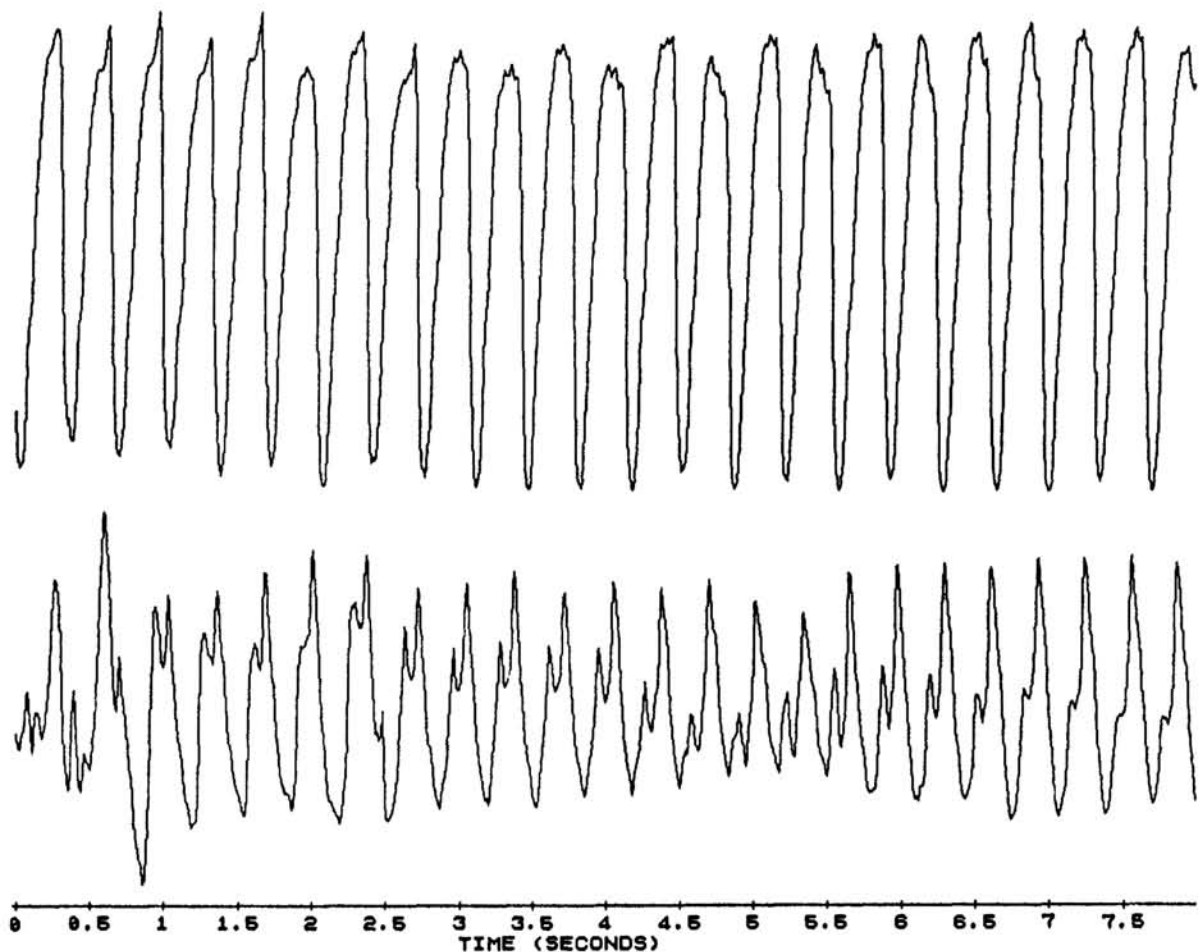

**Figure 6:** Ventricular Tachycardias with Significant Morphology Differences

The misdiagnosis of rhythms not included in the training set can only be corrected by enlarging the training set. In the future, an attempt will be made to create a "generic" set of typical arrhythmias drawn from the entire data set, rather than taking arrhythmia

samples from each patient only. Since the networks can generalize somewhat, it is possible that a network trained on an individual patient's NSR and the "generic" arrhythmia set may be able to recognize all arrhythmias, whether they are included in the training set or not.

## References

C. Giles, R. Griffin, T. Maxwell, "Encoding Geometric Invariances in Higher-Order Neural Networks", Neural Information Processing Systems, American Institute of Physics, New York, 1988, pp.301-309